# A Multi-class Linear Learning Algorithm Related to Winnow

**Chris Mesterharm***
Rutgers Computer Science Department
110 Frelinghuysen Road
Piscataway, NJ 08854
mesterha@paul.rutgers.edu

## Abstract

In this paper, we present Committee, a new multi-class learning algorithm related to the Winnow family of algorithms. Committee is an algorithm for combining the predictions of a set of sub-experts in the on-line mistake-bounded model of learning. A sub-expert is a special type of attribute that predicts with a distribution over a finite number of classes. Committee learns a linear function of sub-experts and uses this function to make class predictions. We provide bounds for Committee that show it performs well when the target can be represented by a few relevant sub-experts. We also show how Committee can be used to solve more traditional problems composed of attributes. This leads to a natural extension that learns on multi-class problems that contain both traditional attributes and sub-experts.

## 1 Introduction

In this paper, we present a new multi-class learning algorithm called Committee. Committee learns a $k$ class target function by combining information from a large set of sub-experts. A sub-expert is a special type of attribute that predicts with a distribution over the target classes. The target space of functions are linear-max functions. We define these as functions that take a linear combination of sub-expert predictions and return the class with maximum value. It may be useful to think of the sub-experts as individual classifying functions that are attempting to predict the target function. Even though the individual sub-experts may not be perfect, Committee attempts to learn a linear-max function that represents the target function. In truth, this picture is not quite accurate. The reason we call them sub-experts and not experts is because even though a individual sub-expert might be poor at prediction, it may be useful when used in a linear-max function. For example, some sub-experts might be used to add constant weights to the linear-max function.

The algorithm is analyzed for the on-line mistake-bounded model of learning [Lit89]. This is a useful model for a type of incremental learning where an algorithm can use feedback about its current hypothesis to improve its performance. In this model, the algorithm goes through a series of learning trials. A trial is composed of three steps. First, the algorithm

receives an instance, in this case, the predictions of the sub-experts. Second, the algorithm predicts a label for the instance; this is the global prediction of Committee. And last, the algorithm receives the true label of the instance; Committee uses this information to update its estimate of the target. The goal of the algorithm is to minimize the total number of prediction mistakes the algorithm makes while learning the target.

The analysis and performance of Committee is similar to another learning algorithm, Winnow [Lit89]. Winnow is an algorithm for learning a linear-threshold function that maps attributes in $[0, 1]$ to a binary target. It is an algorithm that is effective when the concept can be represented with a few relevant attributes, irrespective of the behavior of the other attributes. Committee is similar but deals with learning a target that contains only a few relevant sub-experts. While learning with sub-experts is interesting in it's own right, it turns out the distinction between the two tasks is not significant. We will show in section 5 how to transform attributes from $[0, 1]$ into sub-experts. Using particular transformations, Committee is identical to the Winnow algorithms, Balanced and WMA [Lit89]. Furthermore, we can generalize these transformations to handle attribute problems with multi-class targets. These transformations naturally lead to a hybrid algorithm that allows a combination of sub-experts and attributes for multi-class learning problems. This opens up a range of new practical problems that did not easily fit into the previous framework of $[0, 1]$ attributes and binary classification.

## 2  Previous work

Many people have successfully tried the Winnow algorithms on real-world tasks. In the course of their work, they have made modifications to the algorithms to fit certain aspects of their problem. These modifications include multi-class extensions.

For example, [DKR97] use Winnow algorithms on text classification problems. This multi-class problem has a special form; a document can belong to more than one class. Because of this property, it makes sense to learn a different binary classifier for each class. The linear functions are allowed, even desired, to overlap. However, this paper is concerned with cases where this is not possible. For example, in [GR96] the correct spelling of a word must be selected from a set of many possibilities. In this setting, it is more desirable to have the algorithm select a single word.

The work in [GR96] presents many interesting ideas and modifications of the Winnow algorithms. At a minimum, these modification are useful for improving the performance of Winnow on those particular problems. Part of that work also extends the Winnow algorithm to general multi-class problems. While the results are favorable, the contribution of this paper is to give a different algorithm that has a stronger theoretical foundation for customizing a particular multi-class problem.

Blum also works with multi-class Winnow algorithms on the calendar scheduling problem of [MCF+94]. In [Blu95], a modified Winnow is given with theoretical arguments for good performance on certain types of multi-class disjunctions. In this paper, these results are extended, with the new algorithm Committee, to cover a wider range of multi-class linear functions.

Other related theoretical work on multi-class problems includes the regression algorithm $EG^{\pm}$. In [KW97], Kivinen and Warmuth introduce $EG^{\pm}$, an algorithm related to Winnow but used on regression problems. In general, while regression is a useful framework for many multi-class problems, it is not straightforward how to extend regression to the concepts learned by Committee. A particular problem is the inability of current regression techniques to handle 0-1 loss.

# 3 Algorithm

This section of the paper describes the details of Committee. Near the end of the section, we will give a formal statement of the algorithm.

## 3.1 Prediction scheme

Assume there are $n$ sub-experts. Each sub-expert has a positive weight that is used to vote for $k$ different classes; let $w_i$ be the weight of sub-expert $i$. A sub-expert can vote for several classes by spreading its weight with a prediction distribution. For example, if $k = 3$, a sub-expert may give 3/5 of its weight to class 1, 1/5 of its weight to class 2, and 1/5 of its weight to class 3. Let $x_i$ represent this prediction distribution, where $x_i^j$ is the fraction of the weight sub-expert $i$ gives to class $j$. The vote for class $j$ is $\sum_{i=1}^n w_i x_i^j$. Committee predicts the class that has the highest vote. (On ties, the algorithm picks one of the classes involved in the tie.) We call the function computed by this prediction scheme a linear-max function, since it is the maximum class value taken from a linear combination of the sub-expert predictions.

## 3.2 Target function

The goal of Committee is to minimize the number of mistakes by quickly learning sub-expert weights that correctly classify the target function. Assume there exists $\mu$, a vector of nonnegative weights that correctly classifies the target. Notice that $\mu$ can be multiplied by any constant without changing the target. To remove this confusion, we will normalize the weights to sum to 1, i.e., $\sum_{i=1}^n \mu_i = 1$. Let $\zeta(j)$ be the target's vote for class $j$.

$$\zeta(j) = \sum_{i=1}^n \mu_i x_i^j$$

Part of the difficulty of the learning problem is hidden in the target weights. Intuitively, a target function will be more difficult to learn if there is a small difference between the $\zeta$ votes of the correct and incorrect classes. We measure this difficulty by looking at the minimum difference, over all trials, of the vote of the correct label and the vote of the other labels. Assume for trial $t$ that $\rho_t$ is the correct label.

$$\delta = \min_{t \in Trials} \left( \min_{j \neq \rho_t} (\zeta(\rho_t) - \zeta(j)) \right)$$

Because these are the weights of the target, and the target always makes the correct prediction, $\delta > 0$.

One problem with the above assumptions is that they do not allow noise (cases where $\delta \leq 0$). However, there are variations of the analysis that allow for limited amounts of noise [Lit89, Lit91]. Also experimental work [Lit95, LM] shows the family of Winnow algorithms to be much more robust to noise than the theory would predict. Based on the similarity of the algorithm and analysis, and some preliminary experiments, Committee should be able to tolerate some noise.

## 3.3 Updates

Committee only updates on mistakes using multiplicative updates. The algorithm starts by initializing all weights to $1/n$. During the trials, let $\rho$ be the correct label and $\lambda$ be the predicted label of Committee. When $\lambda \neq \rho$ the weight of each sub-expert $i$ is multiplied by $\alpha^{x_i^\rho - x_i^\lambda}$. This corresponds to increasing the weights of the sub-experts who predicted the

correct label instead of the label Committee predicted. The value of $\alpha$ is initialized at the start of the algorithm. The optimal value of $\alpha$ for the bounds depends on $\delta$. Often $\delta$ is not known in advance, but experiments on Winnow algorithms suggest that these algorithms are more flexible, often performing well with a wider range of $\alpha$ values [LM]. Last, the weights are renormalize to sum to 1. While this is not strictly necessary, normalizing has several advantages including reducing the likelyhood of underflow/overflow errors.

### 3.4 Committee code

**Initialization**

$$\forall i \in \{1, \ldots, n\}\, w_i := 1/n.$$
Set $\alpha > 1$.

**Trials**

**Instance** sub-experts $(x_1, \ldots, x_n)$.
**Prediction** $\lambda$ is the first class $c$ such that for all other classes $j$,
$$\sum_{i=1}^{n} w_i x_i^c \geq \sum_{i=1}^{n} w_i x_i^j.$$
**Update** Let $\rho$ be the correct label. If mistake ($\lambda \neq \rho$)
for i:=1 to n
$$w_i := \alpha^{x_i^\rho - x_i^\lambda} w_i.$$
Normalize weights, $\sum_{i=1}^{n} w_i = 1$

### 3.5 Mistake bound

We do not have the space to give the proof for the mistake bound of Committee, but the technique is similar to the proof of the Winnow algorithm, Balanced, given in [Lit89]. For the complete proof, the reader can refer to [Mes99].

**Theorem 1** *Committee makes at most $2 \ln(n) / \delta^2$ mistakes when the target conditions in section 3.2 are satisfied and $\alpha$ is set to $(1 - \delta)^{-1/2}$.*

Surprisingly, this bound does not refer to the number of classes. The effects of larger values of $k$ show up indirectly in the $\delta$ value.

While it is not obvious, this bound shows that Committee performs well when the target can be represented by a small fraction of the sub-experts. Call the sub-experts in the target the relevant sub-experts. Since $\delta$ is a function of the target, $\delta$ only depends on the relevant sub-experts. On the other hand, the remaining sub-experts have a small effect on the bound since they are only represented in the $\ln(n)$ factor. This means that the mistake bound of Committee is fairly stable even when adding a large number of additional sub-experts. In truth, this doesn't mean that the algorithm will have a good bound when there are few relevant sub-experts. In some cases, a small number of sub-experts can give an arbitrarily small $\delta$ value. (This is a general problem with all the Winnow algorithms.) What it does mean is that, given any problem, increasing the number of irrelevant sub-experts will only have a logarithmic effect on the mistake bound.

## 4 Attributes to sub-experts

Often there are no obvious sub-experts to use in solving a learning problem. Many times the only information available is a set of attributes. For attributes in $[0, 1]$, we will show how to use Committee to learn a natural kind of $k$ class target function, a linear machine. To learn this target, we will transform each attribute into $k$ separate sub-experts. We will use some of the same notion as Committee to help understand the transformation.

### 4.1 Attribute target (linear machine)

A linear machine [DH73] is a prediction function that divides the feature space into disjoint convex regions where each class corresponds to one region. The predictions are made by a comparing the value of $k$ different linear functions where each function corresponds to a class.

More formally, assume there are $m - 1$ attributes and $k$ classes. Let $z_i \in [0, 1]$ be attribute $i$. Assume the target function is represented using $k$ linear functions of the attributes. Let $\zeta(j) = \sum_{i=1}^{m} \mu_i^j z_i$ be the linear function for class $j$ where $\mu_i^j$ is the weight of attribute $i$ in class $j$. Notice that we have added one extra attribute. This attribute is set to 1 and is needed for the constant portion of the linear functions. The target function labels an instance with the class of the largest $\zeta$ function. (Ties are not defined.) Therefore, $\zeta(j)$ is similar to the voting function for class $j$ used in Committee.

### 4.2 Transforming the target

One difficulty with these linear functions is that they may have negative weights. Since Committee only allows targets with nonnegative weights, we need transform to an equivalent problem that has nonnegative weights. This is not difficult. Since we are only concerned with the relative difference between the $\zeta$ functions, we are allowed to add any function to the $\zeta$ functions as long as we add it to all $\zeta$ functions. This gives us a simple procedure to remove negative weights. For example, if $\zeta(1) = 3z_1 - 2z_2 + 1z_3 - 4$, we can add $2z_2 + 4$ to every $\zeta$ function to remove the negative weights from $\zeta(1)$. It is straightforward to extend this and remove all negative weights.

We also need to normalize the weights. Again, since only the relative difference between the $\zeta$ functions matter, we can divide all the $\zeta$ functions by any constant. We normalize the weights to sum to 1, i.e., $\sum_{j=1}^{k} \sum_{i=1}^{n} \mu_i^j = 1$. At this point, without loss of generality, assume that the original $\zeta$ functions are nonnegative and normalized.

The last step is to identify a $\delta$ value. We use the same definition of $\delta$ as Committee substituting the corresponding $\zeta$ functions of the linear machine. Assume for trial $t$ that $\rho_t$ is the correct label.

$$\delta = \min_{t \in Trials} \left( \min_{j \neq \rho_t} (\zeta(\rho_t) - \zeta(j)) \right)$$

### 4.3 Transforming the attributes

The transformation works as follows: convert attribute $z_i$ into $k$ sub-experts. Each sub-expert will always vote for one of the $k$ classes with value $z_i$. The target weight for each of these sub-experts is the corresponding target weight of the attribute, label pair in the $\zeta$ functions. Do this for every attribute.

$$z_i \Longrightarrow \mu_i^1 \begin{pmatrix} z_i \\ 0 \\ \vdots \\ 0 \end{pmatrix} + \mu_i^2 \begin{pmatrix} 0 \\ z_i \\ 0 \\ \vdots \end{pmatrix} + \cdots + \mu_i^k \begin{pmatrix} 0 \\ \vdots \\ 0 \\ z_i \end{pmatrix}$$

Notice that we are not using distributions for the sub-expert predictions. A sub-expert's prediction can be converted to a distribution by adding a constant amount to each class prediction. For example, a sub-expert that predicts $z_1 = .7$, $z_2 = 0$, $z_3 = 0$ can be changed to $z_1 = .8$, $z_2 = .1$, $z_3 = .1$ by adding .1 to each class. This conversion does not affect the predicting or updating of Committee.

**Theorem 2** *Committee makes at most* $2\ln(mk)/\delta^2$ *mistakes on a linear machine, as defined in this section, when* $\alpha$ *is set to* $(1-\delta)^{-1/2}$.

**Proof**: The above target transformation creates $mk$ normalized target sub-experts that vote with the same $\zeta$ functions as the linear machine. Therefore, this set of sub-experts has the same $\delta$ value. Plugging these values into the bound for Committee gives the result. $\square$

This transformation provides a simple procedure for solving linear machine problems. While the details of the transformation may look cumbersome, the actual implementation of the algorithm is relatively simple. There is no need to explicitly keep track of the sub-experts. Instead, the algorithm can use a linear machine type representation. Each class keeps a vector of weights, one weight for each attribute. During an update, only the correct class weights and the predicted class weights are changed. The correct class weights are multiplied by $\alpha^{z_i}$; the predicted class weights are multiplied by $\alpha^{-z_i}$.

The above procedure is very similar to the Balanced algorithm from [Lit89], in fact, for $k = 2$, it is identical. A similar transformation duplicates the behavior of the linear-threshold learning version of WMA as given in [Lit89].

$$z_i \implies \begin{pmatrix} z_i \\ 1 - z_i \end{pmatrix}$$

While this transformation shows some advantages for $k = 2$, more research is needed to determine the proper way to generalize to the multi-class case. For both of these transformations, the bounds given in this paper are equivalent (except for a superficial adjustment in the $\delta$ notation of WMA) to the original bounds given in [Lit89].

### 4.4 Combining attributes and sub-experts

These transformations suggest the proper way to do a hybrid algorithm that combines sub-experts and attributes: use the transformations to create new sub-experts from the attributes and combine them with the original sub-experts when running Committee. It may even be desirable to break original sub-experts into attributes and use both in the algorithm because some sub-experts may perform better on certain classes. For example, if it is felt that a sub-expert is particularly good at class 1, we can perform the following transformation.

$$\begin{pmatrix} x_1 \\ x_2 \\ x_3 \end{pmatrix} \implies \begin{pmatrix} x_1 \\ x_2 \\ x_3 \end{pmatrix} \begin{pmatrix} x_1 \\ 0 \\ 0 \end{pmatrix} \begin{pmatrix} 0 \\ x_1 \\ 0 \end{pmatrix} \begin{pmatrix} 0 \\ 0 \\ x_1 \end{pmatrix}$$

Now, instead of using one weight for the whole sub-expert, Committee can also learn based on the sub-expert's performance for the first class. Even if a good target is representable only with the original sub-experts, these additional sub-experts will not have a large effect because of the logarithmic bound. In the same vein, it may be useful to add constant attributes to a set of sub-experts. These add only $k$ extra sub-experts, but allow the algorithm to represent a larger set of target functions.

## 5   Conclusion

In this paper, we have introduced Committee, a multi-class learning algorithm. We feel that this algorithm will be important in practice, extending the range of problems that can be handled by the Winnow family of algorithms. With a solid theoretical foundation, researchers can customize Winnow algorithms to handle various multi-class problems.

Part of this customization includes feature transformations. We show how Committee can handle general linear machine problems by transforming attributes into sub-experts. This suggests a way to do a hybrid learning algorithm that allows a combination of sub-experts and attributes. This same techniques can also be used to add to the representational power on a standard sub-expert problem.

In the future, we plan to empirically test Committee and the feature transformations on real world problems. Part of this testing will include modifying the algorithm to use extra information, that is related to the proof technique [Mes99], in an attempt to lower the number of mistakes. We speculate that adjusting the multiplier to increase the change in progress per trial will be useful for certain types of multi-class problems.

### Acknowledgments

We thank Nick Littlestone for stimulating this work by suggesting techniques for converting the Balanced algorithm to multi-class targets. Also we thank Haym Hirsh, Nick Littlestone and Warren Smith for providing valuable comments and corrections.

## Footnotes

*Part of this work was supported by NEC Research Institute, Princeton, NJ.

## References

[Blu95] Avrim Blum. Empirical support for winnow and weighted-majority algorithms: results on a calendar scheduling domain. In *ML-95*, pages 64–72, 1995.

[DH73] R. O. Duda and P. Hart. *Pattern Classification and Scene Analysis*. Wiley, New York, 1973.

[DKR97] I. Dagan, Y. Karov, and D. Roth. Mistake-driven learning in text categorization. In *EMNLP-97*, pages 55–63, 1997.

[GR96] A. R. Golding and D. Roth. Applying winnow to context-sensitive spelling correction. In *ML-96*, 1996.

[KW97] Jyrki Kivinen and Manfred K. Warmuth. Additive versus exponentiated gradient updates for linear prediction. *Information and Computation*, 132(1):1–64, 1997.

[Lit89] Nick Littlestone. *Mistake bounds and linear-threshold learning algorithms*. PhD thesis, University of California, Santa Cruz, 1989. Technical Report UCSC-CRL-89-11.

[Lit91] Nick Littlestone. Redundant noisy attributes, attribute errors, and linear-threshold learning using winnow. In *COLT-91*, pages 147–156, 1991.

[Lit95] Nick Littlestone. Comparing several linear-threshold learning algorithms on tasks involving superfluous attributes. In *ML-95*, pages 353–361, 1995.

[LM] Nick Littlestone and Chris Mesterharm. A simulation study of winnow and related algorithms. Work in progress.

[MCF+94] T. Mitchell, R. Caruana, D. Freitag, J. McDermott, and D. Zabowski. Experience with a personal learning assistant. *CACM*, 37(7):81–91, 1994.

[Mes99] Chris Mesterharm. A multi-class linear learning algorithm related to winnow with proof. Technical report, Rutgers University, 1999.
